# Fast Bayesian Inference for Non-Conjugate Gaussian Process Regression

**Mohammad Emtiyaz Khan, Shakir Mohamed, and Kevin P. Murphy**
Department of Computer Science, University of British Columbia

## Abstract

We present a new variational inference algorithm for Gaussian process regression with non-conjugate likelihood functions, with application to a wide array of problems including binary and multi-class classification, and ordinal regression. Our method constructs a concave lower bound that is optimized using an efficient fixed-point updating algorithm. We show that the new algorithm has highly competitive computational complexity, matching that of alternative approximate inference methods. We also prove that the use of concave variational bounds provides stable and guaranteed convergence – a property not available to other approaches. We show empirically for both binary and multi-class classification that our new algorithm converges much faster than existing variational methods, and without any degradation in performance.

## 1 Introduction

Gaussian processes (GP) are a popular non-parametric prior for function estimation. For real-valued outputs, we can combine the GP prior with a Gaussian likelihood and perform exact posterior inference in closed form. However, in other cases, such as classification, the likelihood is no longer conjugate to the GP prior, and exact inference is no longer tractable.

Various approaches are available to deal with this intractability. One approach is Markov Chain Monte Carlo (MCMC) techniques [1, 11, 22, 9]. Although this can be accurate, it is often quite slow, and assessing convergence is challenging. There is therefore great interest in deterministic approximate inference methods. One recent approach is the Integrated Nested Laplace Approximation (INLA) [21], which uses numerical integration to approximate the marginal likelihood. Unfortunately, this method is limited to six or fewer hyperparameters, and is thus not suitable for models with a large number of hyperparameters. Expectation propagation (EP) [17] is a popular alternative, and is a method that approximates the posterior distribution by maintaining expectations and iterating until these expectations are consistent for all variables. Although this is fast and accurate for the case of binary classification [15, 18], there are difficulties extending EP to many other cases, such as multi-class classification and parameter learning [24, 13]. In addition, EP is known to have convergence issues and can be numerically unstable.

In this paper, we use a variational approach, where we compute a lower bound to the log marginal likelihood using Jensen's inequality. Unlike EP, this approach does not suffer from numerical issues and convergence problems, and can easily handle multi-class and other likelihoods. This is an active area of research and many solutions have been proposed, see for example, [23, 6, 5, 19, 14]. Unfortunately, most of these methods are slow, since they attempt to solve for the posterior covariance matrix, which has size $O(N^2)$, where $N$ is the number of data points. In [19], a reparameterization was proposed that only requires computing $O(N)$ variational parameters. Unfortunately, this method relies on a non-concave lower bound. In this paper, we propose a new lower bound that is concave, and derive an efficient iterative algorithm for its maximization. Since the original objective is unimodal, we reach the same global optimum as the other methods, but we do so much faster.

$$p(\mathbf{z}|\mathbf{X}, \boldsymbol{\theta}) = \mathcal{N}(\mathbf{z}|\boldsymbol{\mu}, \boldsymbol{\Sigma}) \qquad (1)$$

$$p(\mathbf{y}|\mathbf{z}) = \prod_{n=1}^{N} p(y_n|z_n) \qquad (2)$$

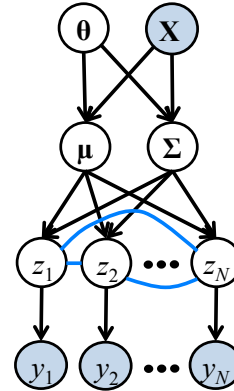

| Type | Distribution | $p(y\|z)$ |
|---|---|---|
| Binary | Bernoulli logit | $p(y=1\|z) = \sigma(z)$ |
| Categorical | Multinomial logit | $p(y=k\|\mathbf{z}) = e^{z_k - \text{lse}(\mathbf{z})}$ |
| Ordinal | Cumulative logit | $p(y \leq k\|z) = \sigma(\phi_k - z)$ |
| Count | Poisson | $p(y=k\|z) = \frac{e^{-e^z} e^{kz}}{k!}$ |

Table 1: Gaussian process regression (top left) and its graphical model (right), along with the example likelihoods for outputs (bottom left). Here, $\sigma(z) = 1/(1 + e^{-z})$, $\text{lse}(\cdot)$ is the log-sum-exp function, $k$ indexes over discrete output values, and $\phi_k$ are real numbers such that $\phi_1 < \phi_2 < \ldots < \phi_K$ for $K$ ordered categories.

## 2 Gaussian Process Regression

Gaussian process (GP) regression is a powerful method for non-parametric regression that has gained a great deal of attention as a flexible and accurate modeling approach. Consider $N$ data points with the $n$'th observation denoted by $y_n$, with corresponding features $\mathbf{x}_n$. A Gaussian process model uses a non-linear latent function $z(\mathbf{x})$ to obtain the distribution of the observation $y$ using an appropriate likelihood [15, 18]. For example, when $y$ is binary, a Bernoulli logit/probit likelihood is appropriate. Similarly, for count observations, a Poisson distribution can be used.

A Gaussian process [20] specifies a distribution over $z(\mathbf{x})$, and is a stochastic process that is characterized by a mean function $\mu(\mathbf{x})$ and a covariance function $\Sigma(\mathbf{x}, \mathbf{x}')$, which are specified using a kernel function that depends on the observed features $\mathbf{x}$. Assuming a GP prior over $z(\mathbf{x})$ implies that a random vector is associated with every input $\mathbf{x}$, such that given all inputs $\mathbf{X} = [\mathbf{x}_1, \mathbf{x}_2, \ldots, \mathbf{x}_N]$, the joint distribution over $\mathbf{z} = [z(\mathbf{x}_1), z(\mathbf{x}_2), \ldots, z(\mathbf{x}_N)]$ is Gaussian.

The GP prior is shown in Eq. 1. Here, $\boldsymbol{\mu}$ is a vector with $\mu(\mathbf{x}_i)$ as its $i$'th element, $\boldsymbol{\Sigma}$ is a matrix with $\Sigma(\mathbf{x}_i, \mathbf{x}_j)$ as the $(i, j)$'th entry, and $\boldsymbol{\theta}$ are the hyperparameters of the mean and covariance functions. We assume throughout a zero mean-function and a squared-exponential covariance function (also known as radial-basis function or Gaussian) defined as: $\Sigma(\mathbf{x}_i, \mathbf{x}_j) = \sigma^2 \exp[-(\mathbf{x}_i - \mathbf{x}_j)^T(\mathbf{x}_i - \mathbf{x}_j)/(2s)]$. The set of hyperparameters is $\boldsymbol{\theta} = (s, \sigma)$. We also define $\boldsymbol{\Omega} = \boldsymbol{\Sigma}^{-1}$.

Given the GP prior, the observations are modeled using the likelihood shown in Eq. 2. The exact form of the distribution $p(y_n|z_n)$ depends on the type of observations and different choices instantiates many existing models for GP regression [15, 18, 10, 14]. We consider frequently encountered data such as binary, ordinal, categorical and count observations, and describe their likelihoods in Table 1. For the case of categorical observations, the latent function $\mathbf{z}$ is a vector whose $k$'th element is the latent function for $k$'th category. A graphical model for Gaussian process regression is also shown.

Given these models, there are three tasks that are to be performed: posterior inference, prediction at test inputs, and model selection. In all cases, the likelihoods we consider are not conjugate to the Gaussian prior distribution and as a result, the posterior distribution is intractable. Similarly, the integrations required in computing the predictive distribution and the marginal likelihood are intractable. To deal with this intractability we make use of variational methods.

## 3 Variational Lower Bound to the Log Marginal Likelihood

Inference and model selection are always problematic in any Gaussian process regression using non-conjugate likelihoods due to the fact that the marginal likelihood contains an intractable integral. In this section, we derive a tractable variational lower bound to the marginal likelihood. We show

that the lower bound takes a well known form and can be maximized using concave optimization. Throughout the section, we assume scalar $z_n$, with extension to the vector case being straightforward.

We begin with the intractable log marginal likelihood $\mathcal{L}(\boldsymbol{\theta})$ in Eq. 3 and introduce a variational posterior distribution $q(\mathbf{z}|\boldsymbol{\gamma})$. We use a Gaussian posterior with mean $\mathbf{m}$ and covariance $\mathbf{V}$. The full set of variational parameters is thus $\boldsymbol{\gamma} = \{\mathbf{m}, \mathbf{V}\}$. As log is a concave function, we obtain a lower bound $\mathcal{L}_J(\boldsymbol{\theta}, \boldsymbol{\gamma})$ using Jensen's inequality, given in Eq. 4. The first integral is simply the Kullback−Leibler (KL) divergence from the variational Gaussian posterior $q(\mathbf{z}|\mathbf{m}, \mathbf{V})$ to the GP prior $p(\mathbf{z}|\boldsymbol{\mu}, \boldsymbol{\Sigma})$ as shown in Eq. 5, and has a closed-form expression that we substitute to get the first term in Eq. 6 (inside square brackets), with $\boldsymbol{\Omega} = \boldsymbol{\Sigma}^{-1}$.

The second integral can be expressed in terms of the expectation with respect to the marginal $q(z_n|m_n, V_{nn})$ as shown in the second term of Eq. 5. Here $m_n$ is the $n$'th element of $\mathbf{m}$ and $V_{nn}$ is the $n$'th diagonal element of $\mathbf{V}$, the two variables collectively denoted by $\gamma_n$. The lower bound $\mathcal{L}_J$ is still intractable since the expectation of $\log p(y_n|z_n)$ is not available in closed form for the distributions listed in Table 1. To derive a tractable lower bound, we make use of local variational bounds (LVB) $f_b$, defined such that $\mathbb{E}[\log p(y_n|z_n)] \geq f_b(y_n, m_n, V_{nn})$, giving us Eq. 6.

$$\mathcal{L}(\boldsymbol{\theta}) = \log \int_{\mathbf{Z}} p(\mathbf{z}|\boldsymbol{\theta})p(\mathbf{y}|\mathbf{z})d\mathbf{z} = \log \int_{\mathbf{Z}} q(\mathbf{z}|\boldsymbol{\gamma})\frac{p(\mathbf{z}|\boldsymbol{\theta})p(\mathbf{y}|\mathbf{z})}{q(\mathbf{z}|\boldsymbol{\gamma})}d\mathbf{z} \tag{3}$$

$$\geq \mathcal{L}_J(\boldsymbol{\theta}, \boldsymbol{\gamma}) := -\int_{\mathbf{Z}} q(\mathbf{z}|\boldsymbol{\gamma}) \log \frac{q(\mathbf{z}|\boldsymbol{\gamma})}{p(\mathbf{z}|\boldsymbol{\theta})}d\mathbf{z} + \int_{\mathbf{Z}} q(\mathbf{z}|\boldsymbol{\gamma}) \log p(\mathbf{y}|\mathbf{z})d\mathbf{z} \tag{4}$$

$$= -D_{KL}\left[q(\mathbf{z}|\boldsymbol{\gamma})||p(\mathbf{z}|\boldsymbol{\theta})\right] + \sum_{n=1}^{N} \mathbb{E}_{q(z_n|\gamma_n)}[\log p(y_n|z_n)] \tag{5}$$

$$\geq \underline{\mathcal{L}}_J(\boldsymbol{\theta}, \boldsymbol{\gamma}) := \tfrac{1}{2}\left[\log|\mathbf{V}\boldsymbol{\Omega}| - \text{tr}(\mathbf{V}\boldsymbol{\Omega}) - (\mathbf{m}-\boldsymbol{\mu})^T\boldsymbol{\Omega}(\mathbf{m}-\boldsymbol{\mu}) + N\right] + \sum_{n=1}^{N} f_b(y_n, m_n, V_{nn}). \tag{6}$$

We discuss the choice of LVBs in the next section, but first discuss the well-known form that the lower bound of Eq. 6 takes. Given $\mathbf{V}$, the optimization function with respect to $\mathbf{m}$ is a nonlinear least-squares function. Similarly, the function with respect to $\mathbf{V}$ is similar to the graphical lasso [8] or covariance selection problem [7], but is different in that the argument is a covariance matrix instead of a precision matrix [8]. These two objective functions are coupled through the non-linear term $f_b(\cdot)$. Usually this term arises due to the prior distribution and may be non-smooth, for example, in graphical lasso. In our case, this term arises from the likelihood, and is smooth and concave as we discuss in next section.

It is straightforward to show that the variational lower bound is strictly concave with respect to $\boldsymbol{\gamma}$ if $f_b$ is jointly concave with respect to $m_n$ and $V_{nn}$. Strict concavity of terms other than $f_b$ is well-known since both the least squares and covariance selection problems are concave. Similar concavity results have been discussed by Braun and McAuliffe [5] for the discrete choice model, and more recently by Challis and Barber [6] for the Bayesian linear model, who consider concavity with respect to the Cholesky factor of $\mathbf{V}$. We consider concavity with respect to $\mathbf{V}$ instead of its Cholesky factor, which allows us to exploit the special structure of $\mathbf{V}$, as explained in Section 5.

## 4  Concave Local Variational Bounds

In this section, we describe concave LVBs for various likelihoods. For simplicity, we suppress the dependence on $n$ and consider the log-likelihood of a scalar observation $y$ given a predictor $z$ distributed according to $q(z|\gamma) = \mathcal{N}(z|m, v)$ with $\gamma = \{m, v\}$. We describe the LVBs for the likelihoods given in Table 1 with $\mathbf{z}$ being a scalar for count, binary, and ordinal data, but a vector of length $K$ for categorical data, $K$ being the number of classes. When $\mathbf{V}$ is a matrix, we denote its diagonal by $\mathbf{v}$.

For the Poison distribution, the expectation is available in closed form and we do not need any bounding: $\mathbb{E}[\log p(y|\eta)] = ym - \exp(m + v/2) - \log y!$. This function is jointly concave with respect to $m$ and $v$ since the exponential is a convex function.

For binary data, we use the piecewise linear/quadratic bounds proposed by [16], which is a bound on the logistic-log-partition (LLP) function $\log(1 + \exp(x))$ and can be used to obtain a bound over the sigmoid function $\sigma(x)$. The final bound can be expressed as sum of $R$ pieces: $\mathbb{E}(\log p(y|\eta)) = f_b(y, m, v) = ym - \sum_{r=1}^{R} f_{br}(m, v)$ where $f_{br}$ is the expectation of $r$'th quadratic piece. The function $f_{br}$ is jointly concave with respect to $m, v$ and their gradients are available in closed-form. An important property of the piecewise bound is that its maximum error is bounded and can be driven to zero by increasing the number of pieces. This means that the lower bound in Eq. 6 can be made arbitrarily tight by increasing the number of pieces. For this reason, this bound always performs better than other existing bounds, such as Jaakola's bound [12], given that the number of pieces is chosen appropriately. Finally, the cumulative logit likelihood for ordinal observations depends on $\sigma(x)$ and its expectation can be bounded using piecewise bounds in a similar way.

For the multinomial logit distribution, we can use the bounds proposed by [3] and [4], both leading to concave LVBs. The first bound takes the form $f_b(y, \mathbf{m}, \mathbf{V}) = \mathbf{y}^T \mathbf{m} - \mathrm{lse}(\mathbf{m} + \mathbf{v}/2)$ with $\mathbf{y}$ represented using a 1-of-K encoding. This function is jointly concave with respect to $\mathbf{m}$ and $\mathbf{v}$, which can be shown by noting the fact that the log-sum-exp function is convex. The second bound is the product of sigmoids bound proposed by [4] which bounds the likelihood with product of sigmoids (see Eq. 3 in [4]), with each sigmoid bounded using Jaakkola's bound [12]. We can also use piecewise linear/quadratic bound to bound each sigmoid. Alternatively, we can use the recently proposed stick-breaking likelihood of [14] which uses piecewise bounds as well.

Finally, note that the original log-likelihood may not be concave itself, but if it is such that $\mathcal{L}_J$ has a unique solution, then designing a concave variational lower bound will allow us to use concave optimization to efficiently maximize the lower bound.

## 5 Existing Algorithms for Variational Inference

In this section, we assume that for each output $y_n$ there is a corresponding scalar latent function $z_n$. All our results can be easily extended to the case of multi-class outputs where the latent function is a vector. In variational inference, we find the approximate Gaussian posterior distribution with mean $\mathbf{m}$ and covariance $\mathbf{V}$ that maximizes Eq. 6. The simplest approach is to use gradient-based methods for optimization, but this can be problematic since the number of variational parameters is quadratic in $N$ due to the covariance matrix $\mathbf{V}$. The authors of [19] speculate that this may perhaps be the reason behind limited use of Gaussian variational approximations.

We now show that the problem is simpler than it appears to be, and in fact the number of parameters can be reduced to $O(N)$ from $O(N^2)$. First, we write the gradients with respect to $\mathbf{m}$ and $\mathbf{v}$ in Eq. 7 and 8 and equate to zero, using $g_n^m := \partial f_b(y_n, m_n, v_n)/\partial m_n$ and $g_n^v := \partial f_b(y_n, m_n, v_n)/\partial v_n$. Also, $\mathbf{g}^m$ and $\mathbf{g}^v$ are the vectors of these gradients, and $\mathrm{diag}(\mathbf{g}^v)$ is the matrix with $\mathbf{g}^v$ as its diagonal.

$$-\mathbf{\Omega}(\mathbf{m} - \boldsymbol{\mu}) + \mathbf{g}^m = 0 \tag{7}$$

$$\tfrac{1}{2}\left(\mathbf{V}^{-1} - \mathbf{\Omega}\right) + \mathrm{diag}(\mathbf{g}^v) = 0 \tag{8}$$

At the solution, we see that $\mathbf{V}$ is completely specified if $\mathbf{g}^v$ is known. This property can be exploited to reduce the number of variational parameters.

Opper and Archambeau [19] (and [18]) propose a reparameterization to reduce the number of parameters to $O(N)$. From the fixed-point equation, we note that at the solution $\mathbf{m}$ and $\mathbf{V}$ will have the following form,

$$\mathbf{V} = (\mathbf{\Sigma}^{-1} + \mathrm{diag}(\boldsymbol{\lambda}))^{-1} \tag{9}$$

$$\mathbf{m} = \boldsymbol{\mu} + \mathbf{\Sigma}\boldsymbol{\alpha}, \tag{10}$$

where $\boldsymbol{\alpha}$ and $\boldsymbol{\lambda}$ are real vectors with $\lambda_d > 0, \forall d$. At the maximum (but not everywhere), $\boldsymbol{\alpha}$ and $\boldsymbol{\lambda}$ will be equal to $\mathbf{g}^m$ and $\mathbf{g}^v$ respectively. Therefore, instead of solving the fixed-point equations to obtain $\mathbf{m}$ and $\mathbf{V}$, we can reparameterize the lower bound with respect to $\boldsymbol{\alpha}$ and $\boldsymbol{\lambda}$. Substituting Eq. 9 and 10 in Eq. 6 and after simplification using the matrix inversion and determinant lemmas, we get the following new objective function (for a detailed derivation, see [18]),

$$\tfrac{1}{2}\left[-\log(|\mathbf{B}_\lambda||\mathrm{diag}(\boldsymbol{\lambda})|) + \mathrm{Tr}(\mathbf{B}_\lambda^{-1}\mathbf{\Sigma}) - \boldsymbol{\alpha}^T\mathbf{\Sigma}\boldsymbol{\alpha}\right] + \sum_{n=1}^{N} f_b(y_n, m_n, V_{nn}), \tag{11}$$

with $\mathbf{B}_\lambda = \text{diag}(\boldsymbol{\lambda})^{-1} + \boldsymbol{\Sigma}$. Since the mapping between $\{\boldsymbol{\alpha}, \boldsymbol{\lambda}\}$ and $\{\mathbf{m}, \mathbf{V}\}$ is one-to-one, we can recover the latter given the former. The one-to-one relationship also implies that the new objective function has a unique maximum. The new lower bound involves vectors of size $N$, reducing the number of variational parameters to $O(N)$.

The problem with this reparameterization is that the new lower bound is no longer concave, even though it has a unique maximum. To see this, consider the 1-D case. We collect all the terms involving $V$ from Eq. 6, except the LVB term, to define the function $f(V) = [\log(V\Sigma^{-1}) - V\Sigma^{-1}]/2$. We substitute the reparameterization $V = (\Sigma^{-1} + \lambda)^{-1}$ to get a new function $f(\lambda) = [-\log(1 + \Sigma\lambda) - (1 + \Sigma\lambda)^{-1}]/2$. The second derivative of this function is $f''(\lambda) = \frac{1}{2}[\Sigma/(1 + \Sigma\lambda)]^2(\Sigma\lambda - 1)$. Clearly, this derivative is negative for $\lambda < 1/\Sigma$ and non-negative otherwise, making the function neither concave nor convex.

The objective function is still unimodal and the maximum of (11) is equal to the maximum of (6). With the reparameterization, we loose concavity and therefore the algorithm may have slow convergence. Our experimental results (Section 7) confirm the slow convergence.

## 6  Fast Convergent Variational Inference using Coordinate Ascent

We now derive an algorithm that reduces the number of variational parameters to $2N$ while maintaining concavity. Our algorithm uses simple scalar fixed-point updates to obtain the diagonal elements of $\mathbf{V}$. The complete algorithm is shown in Algorithm 1.

To derive the algorithm, we first note that the fixed-point equation Eq. 8 has an attractive property: at the solution, the off-diagonal elements of $\mathbf{V}^{-1}$ are the same as the off-diagonal elements of $\boldsymbol{\Omega}$, i.e. if we denote $\mathbf{K} := \mathbf{V}^{-1}$, then $K_{ij} = \Omega_{ij}$. We need only find the diagonal elements of $\mathbf{K}$ to get the full $\mathbf{V}$. This is difficult, however, since the gradient $\mathbf{g}^v$ depends on $\mathbf{v}$.

We take the approach of optimizing each diagonal element $K_{ii}$ fixing all others (and fixing $\mathbf{m}$ as well). We partition $\mathbf{V}$ as shown on the left side of Eq. 12, indexing the last row by 2 and rest of the rows by 1. We consider a similar partitioning of $\mathbf{K}$ and $\boldsymbol{\Omega}$. Our goal is to compute $v_{22}$ and $k_{22}$ given all other elements of $\mathbf{K}$. Matrices $\mathbf{K}$ and $\mathbf{V}$ are related through the blockwise inversion, as shown below.

$$\begin{bmatrix} \mathbf{V}_{11} & \mathbf{v}_{12} \\ \mathbf{v}_{12}^T & v_{22} \end{bmatrix} = \begin{bmatrix} \mathbf{K}_{11}^{-1} + \dfrac{\mathbf{K}_{11}^{-1}\mathbf{k}_{12}\mathbf{k}_{12}^T\mathbf{K}_{11}^{-1}}{k_{22} - \mathbf{k}_{12}^T\mathbf{K}_{11}^{-1}\mathbf{k}_{12}} & -\dfrac{\mathbf{K}_{11}^{-1}\mathbf{k}_{12}}{k_{22} - \mathbf{k}_{12}^T\mathbf{K}_{11}^{-1}\mathbf{k}_{12}} \\ -\dfrac{\mathbf{k}_{12}^T\mathbf{K}_{11}^{-1}}{k_{22} - \mathbf{k}_{12}^T\mathbf{K}_{11}^{-1}\mathbf{k}_{12}} & \dfrac{1}{k_{22} - \mathbf{k}_{12}^T\mathbf{K}_{11}^{-1}\mathbf{k}_{12}} \end{bmatrix} \tag{12}$$

From the right bottom corner, we have the first relation below, which we simplify further.

$$v_{22} = 1/(k_{22} - \mathbf{k}_{12}^T\mathbf{K}_{11}^{-1}\mathbf{k}_{12}) \quad \Rightarrow \quad k_{22} = \widetilde{k}_{22} + 1/v_{22} \tag{13}$$

where we define $\widetilde{k}_{22} := \mathbf{k}_{12}^T\mathbf{K}_{11}^{-1}\mathbf{k}_{12}$. We also know from the fixed point Eq. 8 that the optimal $v_{22}$ and $k_{22}$ satisfy Eq. 14 at the solution, where $g_{22}^v$ is the gradient of $f_b$ with respect to $v_{22}$. Substitute the value of $k_{22}$ from Eq. 13 in Eq. 14 to get Eq. 15. It is easy to check (by taking derivative) that the value $v_{22}$ that satisfies this fixed-point can be found by maximizing the function defined in Eq. 16.

$$0 = k_{22} - \Omega_{22} + 2g_{22}^v \tag{14}$$

$$0 = \widetilde{k}_{22} + 1/v_{22} - \Omega_{22} + 2g_{22}^v \tag{15}$$

$$f(v) = \log(v) - (\Omega_{22} - \widetilde{k}_{22})v + 2f_b(y_2, m_{22}, v) \tag{16}$$

The function $f(v)$ is a strictly concave function and can be optimized by iterating the following update: $v_{22} \leftarrow 1/(\Omega_{22} - \widetilde{k}_{22} - 2g_{22}^v)$. We will refer to this as a "fixed-point iteration".

Since all elements of $\mathbf{K}$, except $k_{22}$, are fixed, $\widetilde{k}_{22}$ can be computed beforehand and need not be evaluated at every fixed-point iteration. In fact, we do not need to compute it explicitly, since we can obtain its value using Eq. 13: $\widetilde{k}_{22} = k_{22} - 1/v_{22}$, and we do this before starting a fixed-point iteration. The complexity of these iterations depends on the number of gradient evaluations $g_{22}^v$, which is usually constant and very low.

After convergence of the fixed-point iterations, we update $\mathbf{V}$ using Eq. 12. It turns out that this is a rank-one update, the complexity of which is $O(N^2)$. To show these updates, let us denote the new values obtained after the fixed-point iterations by $k_{22}^{new}$ and $v_{22}^{new}$ respectively. and denote the old values by $k_{22}^{old}$ and $v_{22}^{old}$. We use the right top corner of Eq. 12 to get first equality in Eq. 17. Using Eq. 13, we get the second equality. Similarly, we use the top left corner of Eq. 12 to get the first equality in Eq. 18, and use Eq. 13 and 17 to get the second equality.

$$\mathbf{K}_{11}^{-1}\mathbf{k}_{12} = -(k_{22}^{old} - \widetilde{k}_{22})\mathbf{v}_{12}^{old} = -\mathbf{v}_{12}^{old}/v_{22}^{old} \tag{17}$$

$$\mathbf{K}_{11}^{-1} = \mathbf{V}_{11}^{old} - \frac{\mathbf{K}_{11}^{-1}\mathbf{k}_{12}\mathbf{k}_{12}^{T}\mathbf{K}_{11}^{-1}}{k_{22}^{old} - \widetilde{k}_{22}} = \mathbf{V}_{11}^{old} - \mathbf{v}_{12}^{old}(\mathbf{v}_{12}^{old})^{T}/v_{22}^{old} \tag{18}$$

Note that both $\mathbf{K}_{11}^{-1}$ and $\mathbf{k}_{12}$ do not change after the fixed point iteration. We use this fact to obtain $\mathbf{V}^{new}$. We use Eq. 12 to write updates for $\mathbf{V}^{new}$ and use 17, 18, and 13 to simplify.

$$\mathbf{v}_{12}^{new} = \frac{\mathbf{K}_{11}^{-1}\mathbf{k}_{12}}{k_{22}^{new} - \widetilde{k}_{22}} = -\frac{v_{22}^{new}}{v_{22}^{old}}\mathbf{v}_{12}^{old} \tag{19}$$

$$\mathbf{V}_{11}^{new} = \mathbf{K}_{11}^{-1} + \frac{\mathbf{K}_{11}^{-1}\mathbf{k}_{12}\mathbf{k}_{12}^{T}\mathbf{K}_{11}^{-1}}{k_{22}^{new} - \widetilde{k}_{22}} = \mathbf{V}_{11}^{old} + \frac{v_{22}^{new} - v_{22}^{old}}{(v_{22}^{old})^2}\mathbf{v}_{12}^{old}(\mathbf{v}_{12}^{old})^{T} \tag{20}$$

After updating $\mathbf{V}$, we update $\mathbf{m}$ by optimizing the following non-linear least squares problem,

$$\max_{\mathbf{m}} -\tfrac{1}{2}(\mathbf{m} - \boldsymbol{\mu})^{T}\boldsymbol{\Omega}(\mathbf{m} - \boldsymbol{\mu}) + \sum_{n=1}^{N} f_b(y_n, m_n, V_{nn}) \tag{21}$$

We use Newton's method, the cost of which is $O(N^3)$.

## 6.1 Computational complexity

The final procedure is shown in Algorithm 1. The main advantage of our algorithm is its fast convergence as we show this in the results section. The overall computational complexity is $O(N^3 + \sum_n I_n^{fp})$. First term is due to $O(N^2)$ update of $\mathbf{V}$ for all $n$ and also due to the optimization of $\mathbf{m}$. Second term is for $I_n^{fp}$ fixed-point iterations, the total cost of which is linear in $N$ due to the summation. In all our experiments, $I_n^{fp}$ is usually 3 to 5, adding very little cost.

## 6.2 Proof of convergence

Proposition 2.7.1 in [2] states that the coordinate ascent algorithm converges if the maximization with respect to each coordinate is uniquely attained. This is indeed the case for us since each fixed point iteration solves a concave problem of the form given by Eq. 16. Similarly, optimization with respect to $\mathbf{m}$ is also strictly concave. Hence, convergence of our algorithm is assured.

## 6.3 Proof that V will always be positive definite

Let us assume that we start with a positive definite $\mathbf{K}$, for example, we can initialize it with $\boldsymbol{\Omega}$. Now consider the update of $v_{22}$ and $k_{22}$. Note that $v_{22}^{new}$ will be positive since it is the maximum of Eq. 16 which involves the log term. Using this and Eq. 13, we get $k_{22}^{new} > \mathbf{k}_{12}^{T}\mathbf{K}_{11}^{-1}\mathbf{k}_{12}$. Hence, the Schur complement $k_{22}^{new} - \mathbf{k}_{12}^{T}\mathbf{K}_{11}^{-1}\mathbf{k}_{12} > 0$. Using this and the fact that $\mathbf{K}_{11}$ is positive definite, it follows that $\mathbf{K}^{new}$ will also be positive definite, and hence $\mathbf{V}^{new}$ will be positive definite.

# 7 Results

We now show that the proposed algorithm leads to a significant gain in the speed of Gaussian process regression. The software to reproduce the results of this section are available online[1]. We evaluate the performance of our fast variational inference algorithm against existing inference methods for

**Algorithm 1** Fast convergent coordinate-ascent algorithm
---
   1. Initialize $\mathbf{K} \leftarrow \mathbf{\Omega}, \mathbf{V} \leftarrow \mathbf{\Omega}^{-1}, \mathbf{m} \leftarrow \boldsymbol{\mu}$, where $\mathbf{\Omega} := \mathbf{\Sigma}^{-1}$.

   2. Alternate between updating the diagonal of $\mathbf{V}$ and then $\mathbf{m}$ until convergence, as follows:

      (a) Update the $i$'th diagonal of $\mathbf{V}$ for all $i = 1, \ldots, N$:

         i. Rearrange $\mathbf{V}$ and $\mathbf{\Omega}$ so that the $i$'th column is the last one.

         ii. $\widetilde{k}_{22} \leftarrow k_{22} - 1/v_{22}$.

         iii. Store old value $v_{22}^{old} \leftarrow v_{22}$.

         iv. Run fixed-point iterations for a few steps: $v_{22} \leftarrow 1/(\Omega_{22} - \widetilde{k}_{22} - 2g_{22}^v)$.

         v. Update $\mathbf{V}$.

           A. $\mathbf{V}_{11} \leftarrow \mathbf{V}_{11} + (v_{22} - v_{22}^{old})\mathbf{v}_{12}\mathbf{v}_{12}^T/(v_{22}^{old})^2$.

           B. $\mathbf{v}_{12} \leftarrow -v_{22}\mathbf{v}_{12}/v_{22}^{old}$.

         vi. Update $k_{22} \leftarrow \widetilde{k}_{22} + 1/v_{22}$.

      (b) Update $\mathbf{m}$ by maximizing the least-squares problem of Eq. 21.
---

binary and multi-class classification. For binary classification, we use the UCI ionosphere data (with 351 data examples containing 34 features). For multi-class classification, we use the UCI forensic glass data set with 214 data examples each with 6 category output and features of length 8. In both cases, we use 80% of the dataset for training and the rest for testing.

We consider *GP classification* using the Bernoulli logit likelihood, for which we use the piecewise bound of [16] with 20 pieces. We compare our algorithm with the approach of Opper and Archambeau [19] (Eq. 11). For the latter, we use L-BFGS method for optimization. We also compared to the naive method of optimizing with respect to full $\mathbf{m}$ and $\mathbf{V}$, e.g. method of [5], but do not present these results since these algorithms have very slow convergence.

We examine the computational cost for each method in terms of the number of floating point operations (flops) for four hyperparameter settings $\boldsymbol{\theta} = \{\log(s), \log(\sigma)\}$. This comparison is shown in Figure 1(a). The y-axis shows (negative of) the value of the lower bound, and the x-axis shows the number of flops. We draw markers at iteration 1,2,4,50 and in steps of 50 from then on. In all cases, due to non-concavity, the optimization of the Opper and Archambeau reparameterization (black curve with squares) convergence slowly, passing through flat regions of the objective and requiring a large number of computations to reach convergence. The proposed algorithm (blue curve with circles) has consistently faster convergence than the existing method. For this dataset, our algorithm always converged in 5 iterations.

We also compare the total cost to convergence, where we count the total number of flops until successive increase in the objective function is below $10^{-3}$. Each entry is a different setting of $\{\log(s), \log(\sigma)\}$. Rows correspond to values of $\log(s)$ while columns correspond to $\log(\sigma)$, with units M,G,T denoting Mega-, Giga-, and Terra-flops. We can see that the proposed algorithm takes a much smaller number of operations compared to the existing algorithm.

<table>
<tr><td colspan="4" align="center">**Proposed Algorithm**</td><td colspan="4" align="center">**Opper and Archambeau**</td></tr>
<tr><td></td><td>-1</td><td>1</td><td>3</td><td></td><td>-1</td><td>1</td><td>3</td></tr>
<tr><td>-1</td><td>6M</td><td>7M</td><td>7M</td><td>-1</td><td>20G</td><td>212G</td><td>6T</td></tr>
<tr><td>1</td><td>26M</td><td>20M</td><td>22M</td><td>1</td><td>101G</td><td>24T</td><td>24T</td></tr>
<tr><td>3</td><td>47M</td><td>81M</td><td>75M</td><td>3</td><td>38G</td><td>1T</td><td>24T</td></tr>
</table>

We also applied our method to two more datasets of [18], namely 'sonar' and 'usps-3vs5' dataset and observed similar behavior.

Next, we apply our algorithm to the problem of *multi-class classification*, following [14], using the stick-breaking likelihood, and compare to inference using the approach of Opper and Archambeau [19] (Eq. 11). We show results comparing the lower bound vs the number of flops taken in Figure 1(b), for four hyperparameter settings $\{\log(s), \log(\sigma)\}$. We show markers at iterations 1, 2, 10, 100 and every 100th iteration thereafter. The results follow those discussed for binary classification,

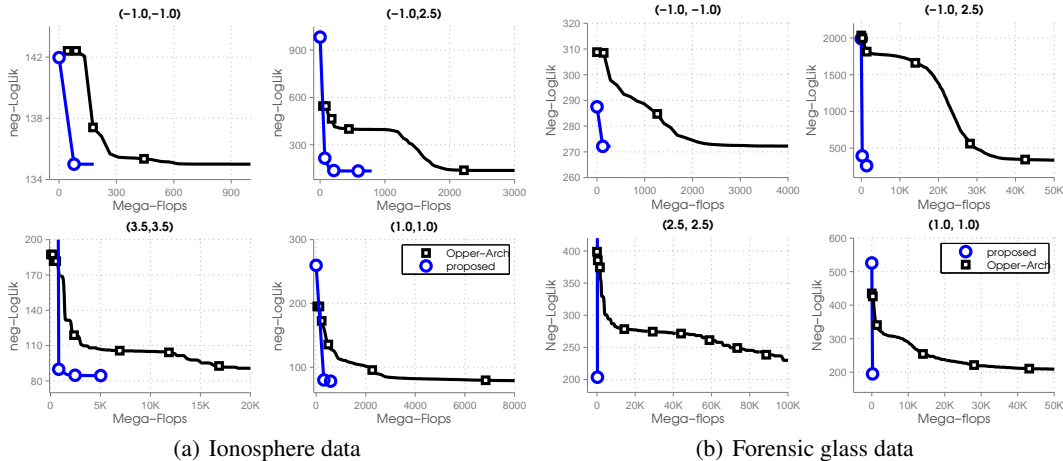

(a) Ionosphere data                    (b) Forensic glass data

Figure 1: Convergence results for (a) the binary classification on the ionosphere data set and (b) the multi-class classification on the glass dataset. We plot the negative of the lower bound vs the number of flops. Each plot shows the progress of algorithms for a hyperparameter setting $\{\log(s), \log(\sigma)\}$ shown at the top of the plot. The proposed algorithm always converges faster than the other method, in fact, in less than 5 iterations.

where both methods reach the same lower bound value, but the existing approach converging much slower, with our algorithm always converged within 20 iterations.

# 8  Discussion

In this paper we have presented a new variational inference algorithm for non-conjugate GP regression. We derived a concave variational lower bound to the log marginal likelihood, and used concavity to develop an efficient optimization algorithm. We demonstrated the efficacy of our new algorithm on both binary and multiclass GP classification, demonstrating significant improvement in convergence.

Our proposed algorithm is related to many existing methods for GP regression. For example, the objective function that we consider is exactly the KL minimization method discussed in [18], for which a gradient based optimization was used. Our algorithm uses an efficient approach where we update the marginals of the posterior and then do a rank one update of the covariance matrix. Our results show that this leads to fast convergence.

Our algorithm also takes a similar form to the popular EP algorithm [17], e.g. see Algorithm 3.5 in [20]. Both EP and our algorithm update posterior marginals, followed by a rank-one update of the covariance. Therefore, the computational complexity of our approach is similar to that of EP. The advantage of our approach is that, unlike EP, it does not suffer from any numerical issues (for example, no negative variances) and is guaranteed to converge.

The derivation of our algorithm is based on the observation that the posterior covariance has a special structure, and does not directly use the concavity of the lower bound. An alternate derivation based on the Fenchel duality exists and shows that the fixed-point iterations compute dual variables which are related to the gradients of $f_b$. We skip this derivation since it is tedious, and present the more intuitive derivation instead. The alternative derivation will be made available in an online appendix.

### Acknowledgements

We thank the reviewers for their valuable suggestions. SM is supported by the Canadian Institute for Advanced Research (CIFAR).

## Footnotes

[1] http://www.cs.ubc.ca/emtiyaz/software/codeNIPS2012.html

# References

[1] J. Albert and S. Chib. Bayesian analysis of binary and polychotomous response data. *J. of the Am. Stat. Assoc.*, 88(422):669–679, 1993.

[2] Dimitri P. Bertsekas. *Nonlinear Programming*. Athena Scientific, second edition, 1999.

[3] D. Blei and J. Lafferty. Correlated topic models. In *Advances in Neural Information Proceedings Systems*, 2006.

[4] G. Bouchard. Efficient bounds for the softmax and applications to approximate inference in hybrid models. In *NIPS 2007 Workshop on Approximate Inference in Hybrid Models*, 2007.

[5] M. Braun and J. McAuliffe. Variational inference for large-scale models of discrete choice. *Journal of the American Statistical Association*, 105(489):324–335, 2010.

[6] E. Challis and D. Barber. Concave Gaussian variational approximations for inference in large-scale Bayesian linear models. In *Proceedings of the International Conference on Artificial Intelligence and Statistics*, volume 6, page 7, 2011.

[7] A. Dempster. Covariance selection. *Biometrics*, 28(1), 1972.

[8] J. Friedman, T. Hastie, and R. Tibshirani. Sparse inverse covariance estimation with the graphical lasso. *Biostatistics*, 9(3):432, 2008.

[9] S. Frühwirth-Schnatter and R. Frühwirth. Data augmentation and MCMC for binary and multinomial logit models. *Statistical Modelling and Regression Structures*, pages 111–132, 2010.

[10] M. Girolami and S. Rogers. Variational Bayesian multinomial probit regression with Gaussian process priors. *Neural Comptuation*, 18(8):1790 – 1817, 2006.

[11] C. Holmes and L. Held. Bayesian auxiliary variable models for binary and multinomial regression. *Bayesian Analysis*, 1(1):145–168, 2006.

[12] T. Jaakkola and M. Jordan. A variational approach to Bayesian logistic regression problems and their extensions. In *AI + Statistics*, 1996.

[13] P. Jylänki, J. Vanhatalo, and A. Vehtari. Robust Gaussian process regression with a student-t likelihood. *The Journal of Machine Learning Research*, 999888:3227–3257, 2011.

[14] M. Khan, S. Mohamed, B. Marlin, and K. Murphy. A stick-breaking likelihood for categorical data analysis with latent Gaussian models. In *Proceedings of the International Conference on Artificial Intelligence and Statistics*, 2012.

[15] M. Kuss and C. E. Rasmussen. Assessing approximate inference for binary Gaussian process classification. *J. of Machine Learning Research*, 6:1679–1704, 2005.

[16] B. Marlin, M. Khan, and K. Murphy. Piecewise bounds for estimating Bernoulli-logistic latent Gaussian models. In *Intl. Conf. on Machine Learning*, 2011.

[17] T. Minka. Expectation propagation for approximate Bayesian inference. In *UAI*, 2001.

[18] H. Nickisch and C.E. Rasmussen. Approximations for binary Gaussian process classification. *Journal of Machine Learning Research*, 9(10), 2008.

[19] M. Opper and C. Archambeau. The variational Gaussian approximation revisited. *Neural computation*, 21(3):786–792, 2009.

[20] C. E. Rasmussen and C. K. I. Williams. *Gaussian Processes for Machine Learning*. MIT Press, 2006.

[21] H. Rue, S. Martino, and N. Chopin. Approximate Bayesian inference for latent Gaussian models using integrated nested Laplace approximations. *J. of Royal Stat. Soc. Series B*, 71:319–392, 2009.

[22] S. L. Scott. Data augmentation, frequentist estimation, and the Bayesian analysis of multinomial logit models. *Statistical Papers*, 52(1):87–109, 2011.

[23] M. Seeger. Bayesian Inference and Optimal Design in the Sparse Linear Model. *J. of Machine Learning Research*, 9:759–813, 2008.

[24] M. Seeger and H. Nickisch. Fast Convergent Algorithms for Expectation Propagation Approximate Bayesian Inference. In *Proceedings of the International Conference on Artificial Intelligence and Statistics*, 2011.

